# Bayesian Experimental Design of Magnetic Resonance Imaging Sequences

**Matthias W. Seeger, Hannes Nickisch, Rolf Pohmann and Bernhard Schölkopf**
Max Planck Institute for Biological Cybernetics
Spemannstraße 38
72012 Tübingen, Germany
{seeger,hn,rolf.pohmann,bs}@tuebingen.mpg.de

## Abstract

We show how improved sequences for magnetic resonance imaging can be found through optimization of Bayesian design scores. Combining approximate Bayesian inference and natural image statistics with high-performance numerical computation, we propose the first Bayesian experimental design framework for this problem of high relevance to clinical and brain research. Our solution requires large-scale approximate inference for dense, non-Gaussian models. We propose a novel scalable variational inference algorithm, and show how powerful methods of numerical mathematics can be modified to compute primitives in our framework. Our approach is evaluated on raw data from a 3T MR scanner.

## 1 Introduction

Magnetic resonance imaging (MRI) [7, 2] is a key diagnostic technique in healthcare nowadays, and of central importance for experimental research of the brain. Without applying any harmful ionizing radiation, this technique stands out by its amazing versatility: by combining different types of radiofrequency irradiation and rapidly switched spatially varying magnetic fields (called *gradients*) superimposing the homogeneous main field, a large variety of different parameters can be recorded, ranging from basic anatomy to imaging blood flow, brain function or metabolite distribution. For this large spectrum of applications, a huge number of *sequences* has been developed that describe the temporal flow of the measurement, ranging from a relatively low number of multi-purpose techniques like FLASH [5], RARE [6], or EPI [9], to specialized methods for visualizing bones or perfusion. To select the optimum sequence for a given problem, and to tune its parameters, is a difficult task even for experts, and even more challenging is the design of new, customized sequences to address a particular question, making sequence development an entire field of research [1]. The main drawbacks of MRI are high initial and running costs, since a very strong homogeneous magnetic field has to be maintained, moreover long scanning times due to weak signals and limits to gradient amplitude. With this in mind, by far the majority of scientific work on improving MRI is motivated by obtaining diagnostically useful images in less time. Beyond reduced costs, faster imaging also leads to higher temporal resolution in dynamic sequences for functional MRI (fMRI), less annoyance to patients, and fewer artifacts due to patient motion.

In this paper, we employ *Bayesian experimental design* to optimize MRI sequences. Image reconstruction from MRI raw data is viewed as a problem of inference from incomplete observations. In contrast, current reconstruction techniques are non-iterative. For most sequences used in hospitals today, reconstruction is done by a single fast Fourier transform (FFT). However, natural and MR images show stable low-level statistical properties,[1] which allows them to be reconstructed from

fewer observations. In our work, a non-Gaussian prior distribution represents low-level spectral and local natural image statistics. A similar idea is known as *compressed sensing* (CS), which has been applied to MRI [8].

A different and more difficult problem is to improve the sequence itself. In our Bayesian method, a posterior distribution over images is maintained, which is *essential* for judging the quality of the sequence: the latter can be modified so as to decrease uncertainty in regions or along directions of interest, where uncertainty is quantified by the posterior. Importantly, this is done without the need to run many MRI experiments in random a priori data collections. It has been proposed to design sequences by blindly randomizing aspects thereof [8], based on CS theoretical results. Beyond being hard to achieve on a scanner, our results indicate that random measurements do not work well for real MR images. Similar negative findings for a variety of natural images are given in [12].

Our proposal requires efficient Bayesian inference for MR images of realistic resolution. We present a novel scalable variational approximate inference algorithm inspired by [16]. The problem is reduced to numerical mathematics primitives, and further to matrix-vector multiplications (MVM) with large, structured matrices, which are computed by efficient signal processing code. Most previous algorithms [3, 14, 11] iterate over single non-Gaussian potentials, which renders them of no use for our problem here.[2] Our solutions for primitives required here should be useful for other machine learning applications as well. Finally, we are not aware of Bayesian or classical experimental design methods for dense *non-Gaussian* models, scaling comparably to ours. The framework of [11] is similar, but could not be applied to the scale of interest here. Our model and experimental design framework are described in Section 2, a novel scalable approximate inference algorithm is developed in Section 3, and our framework is evaluated on a large-scale realistic setup with scanner raw data in Section 4.

## 2  Sparse Linear Model. Experimental Design

Denote the desired MR image by $\boldsymbol{u} \in \mathbb{R}^n$, where $n$ is the number of pixels. Under ideal conditions, the raw data $\boldsymbol{y} \in \mathbb{R}^m$ from the scanner is a linear map[3] of $\boldsymbol{u}$, motivating the likelihood

$$\boldsymbol{y} = \boldsymbol{X}\boldsymbol{u} + \boldsymbol{\varepsilon}, \quad \boldsymbol{\varepsilon} \sim N(\boldsymbol{0}, \sigma^2 \boldsymbol{I}).$$

Here, each row of $\boldsymbol{X}$ is a single Fourier filter, determined by the sequence. In the context of this paper, the problem of *experimental design* is how to choose $\boldsymbol{X}$ within a space of technically feasible sequences, so that $\boldsymbol{u}$ can be best recovered given $\boldsymbol{y}$. As motivated in Section 1, we need to specify a prior $P(\boldsymbol{u})$ which represents low-level statistics of (MR) images, distinctly super-Gaussian distributions — a Gaussian prior would not be a sensible choice. We use the one proposed in [12]. The posterior has the form

$$P(\boldsymbol{u}|\boldsymbol{y}) \propto N(\boldsymbol{y}|\boldsymbol{X}\boldsymbol{u}, \sigma^2 \boldsymbol{I}) \prod_{j=1}^{q} e^{-\tilde{\tau}_j |s_j|}, \quad \boldsymbol{s} = \boldsymbol{B}\boldsymbol{u}, \ \tilde{\tau}_j = \tau_j / \sigma, \tag{1}$$

the prior being a product of Laplacians on linear projections $s_j$ of $\boldsymbol{u}$, among them the image gradient and wavelet coefficients. The Laplace distribution encourages sparsity of $\boldsymbol{s}$. Further details are given in [12]. MVMs with $\boldsymbol{B}$ cost $O(q)$ with $q \approx 3n$. MAP estimation for the same model was used in [8].

Bayesian inference for (1) is analytically not tractable, and an efficient deterministic approximation is discussed in Section 3. In the variant of Bayesian sequential experimental design used here, an extension of $\boldsymbol{X}$ by $\boldsymbol{X}_* \in \mathbb{R}^{d,n}$ is scored by the *entropy difference*

$$\Delta(\boldsymbol{X}_*) := \mathrm{H}[P(\boldsymbol{u}|\boldsymbol{y})] - \mathrm{E}_{P(\boldsymbol{y}_*|\boldsymbol{y})} \left[ \mathrm{H}[P(\boldsymbol{u}|\boldsymbol{y}, \boldsymbol{y}_*)] \right], \tag{2}$$

where $P(\boldsymbol{u}|\boldsymbol{y}, \boldsymbol{y}_*)$ is the posterior after including $(\boldsymbol{X}_*, \boldsymbol{y}_*)$. This criterion measures the decrease in uncertainty about $\boldsymbol{u}$, averaged over the posterior $P(\boldsymbol{y}_*|\boldsymbol{y})$. Our approach is sequential: a sequence is combined from parts, each extension being chosen by maximizing the entropy difference over a

candidate set $\{\boldsymbol{X}_*\}$. After each extension, a new scanner measurement is obtained for the single extended sequence only. Our Bayesian predictive approach allows us to score many candidates $(\boldsymbol{X}_*, \boldsymbol{y}_*)$ without performing costly MR measurements for them. The sequential restriction makes sense for several reasons. First, MR sequences naturally decompose in a sequential fashion: they describe a discontinuous path of several smooth trajectories (see Section 4). Also, a non-sequential approach would never make use of any real measurements, relying much more on the correctness of the model. Finally, the computational complexity of optimizing over complete sequences is staggering. Our sequential approach seems also better suited for dynamic MRI applications.

## 3 Scalable Approximate Inference

In this section, we propose a novel scalable algorithm for the variational inference approximation proposed in [3]. We make use of ideas presented in [16]. First, $e^{-\tilde{\tau}_j|s_j|} = \max_{\pi_j > 0} e^{-\pi_j s_j^2/(2\sigma^2)} e^{-(\tau_j^2/2)\pi_j^{-1}}$, using Legendre duality (the Laplace site is log-convex in $s_j^2$) [3]. Let $\boldsymbol{\pi} = (\pi_j)$ and $\boldsymbol{\Pi} = \operatorname{diag} \boldsymbol{\pi}$. To simplify the derivation, assume that $\boldsymbol{B}^T \boldsymbol{\Pi} \boldsymbol{B}$ is invertible,[4] and let $Q(\boldsymbol{u}) \propto \exp(-\boldsymbol{u}^T \boldsymbol{B}^T \boldsymbol{\Pi} \boldsymbol{B} \boldsymbol{u}/(2\sigma^2))$, $Q(\boldsymbol{y}, \boldsymbol{u}) := P(\boldsymbol{y}|\boldsymbol{u})Q(\boldsymbol{u})$. The joint distribution is Gaussian, and

$$Q(\boldsymbol{u}|\boldsymbol{y}) = N(\boldsymbol{u}|\boldsymbol{h}, \sigma^2 \boldsymbol{\Sigma}), \quad \boldsymbol{\Sigma}^{-1} = \boldsymbol{A} := \boldsymbol{X}^T \boldsymbol{X} + \boldsymbol{B}^T \boldsymbol{\Pi} \boldsymbol{B}, \ \boldsymbol{h} = \boldsymbol{\Sigma} \boldsymbol{X}^T \boldsymbol{y}. \quad (3)$$

We have that $P(\boldsymbol{y}) \geq e^{-\frac{1}{2}(\boldsymbol{\tau}^2)^T(\boldsymbol{\pi}^{-1})}|\boldsymbol{B}^T \boldsymbol{\Pi} \boldsymbol{B}/(2\pi\sigma^2)|^{-1/2} \int P(\boldsymbol{y}|\boldsymbol{u})Q(\boldsymbol{u}) \, d\boldsymbol{u}$, and

$$\int P(\boldsymbol{y}|\boldsymbol{u})Q(\boldsymbol{u}) \, d\boldsymbol{u} = |2\pi\sigma^2 \boldsymbol{\Sigma}|^{1/2} \max_{\boldsymbol{u}} Q(\boldsymbol{u}|\boldsymbol{y})Q(\boldsymbol{y}) = |2\pi\sigma^2 \boldsymbol{\Sigma}|^{1/2} \max_{\boldsymbol{u}} P(\boldsymbol{y}|\boldsymbol{u})Q(\boldsymbol{u}),$$

where the maximum is attained at $\boldsymbol{u} = \boldsymbol{h}$. Therefore, $P(\boldsymbol{y}) \geq C_1(\sigma^2)e^{-\phi(\boldsymbol{\pi})/2}$ with

$$\phi(\boldsymbol{\pi}) := \log|\boldsymbol{A}| + (\boldsymbol{\tau}^2)^T(\boldsymbol{\pi}^{-1}) + \min_{\boldsymbol{u}} \sigma^{-2}\|\boldsymbol{y} - \boldsymbol{X}\boldsymbol{u}\|^2 + \sigma^{-2}\boldsymbol{s}^T \boldsymbol{\Pi} \boldsymbol{s}, \quad \boldsymbol{s} = \boldsymbol{B}\boldsymbol{u},$$

and the bound is tightened by minimizing $\phi(\boldsymbol{\pi})$. Now, $g(\boldsymbol{\pi}) := \log|\boldsymbol{A}|$ is concave, so we can use another Legendre duality, $g(\boldsymbol{\pi}) = \min_{\boldsymbol{z} \succeq \boldsymbol{0}} \boldsymbol{z}^T \boldsymbol{\pi} - g^*(\boldsymbol{z})$, to obtain an upper bound $\phi_{\boldsymbol{z}}(\boldsymbol{\pi}) = \min_{\boldsymbol{u}} \phi_{\boldsymbol{z}}(\boldsymbol{u}, \boldsymbol{\pi}) \geq \phi(\boldsymbol{\pi})$. In the outer loop steps of our algorithm, we need to find the minimizer $\boldsymbol{z} \in \mathbb{R}_+^q$; the inner loop consists of minimizing the upper bound w.r.t. $\boldsymbol{\pi}$ for fixed $\boldsymbol{z}$. Introducing $\boldsymbol{\gamma} := \boldsymbol{\pi}^{-1}$, we find that $(\boldsymbol{u}, \boldsymbol{\gamma}) \mapsto \phi_{\boldsymbol{z}}(\boldsymbol{u}, \boldsymbol{\gamma}^{-1})$ is *jointly* convex, which follows just as in [16], and because $\boldsymbol{z}^T(\boldsymbol{\gamma}^{-1})$ is convex (all $z_j \geq 0$). Minimizing over $\boldsymbol{\gamma}$ gives the convex problem

$$\min_{\boldsymbol{u}} \sigma^{-2}\|\boldsymbol{y} - \boldsymbol{X}\boldsymbol{u}\|^2 + 2\sum_{j=1}^q \tau_j \sqrt{p_j}, \quad p_j := z_j + \sigma^{-2}s_j^2, \ \boldsymbol{s} = \boldsymbol{B}\boldsymbol{u}, \quad (4)$$

which is of standard form and can be solved very efficiently by the *iteratively reweighted least squares* (IRLS) algorithm, a special case of Newton-Raphson. In every iteration, we have to solve $(\boldsymbol{X}^T \boldsymbol{X} + \boldsymbol{B}^T(\operatorname{diag} \boldsymbol{e})\boldsymbol{B})\boldsymbol{d} = \boldsymbol{r}$, where $\boldsymbol{r}, \boldsymbol{e}$ are simple functions of $\boldsymbol{u}$. We use the linear conjugate gradients (LCG) algorithm [4], requiring a MVM with $\boldsymbol{X}, \boldsymbol{X}^T, \boldsymbol{B}$, and $\boldsymbol{B}^T$ per iteration. The line search along the Newton direction $\boldsymbol{d}$ can be done in $O(q)$, no further MVMs are required. In our experiments, IRLS converged rapidly. At convergence, $\pi_j' = \tau_j(p_j')^{-1/2}$, $\boldsymbol{p}' = \boldsymbol{p}'(\boldsymbol{u}')$. For updating $\boldsymbol{z} \to \boldsymbol{z}'$ given $\boldsymbol{\pi}$, note that $\boldsymbol{\pi}^T \boldsymbol{z}' - g(\boldsymbol{\pi}) = g^*(\boldsymbol{z}') = \min_{\tilde{\boldsymbol{\pi}}} \tilde{\boldsymbol{\pi}}^T \boldsymbol{z}' - g(\tilde{\boldsymbol{\pi}})$, so that $\boldsymbol{0} = \nabla_{\boldsymbol{\pi}} \boldsymbol{\pi}^T \boldsymbol{z}' - g(\boldsymbol{\pi}) = \boldsymbol{z}' - \nabla_{\boldsymbol{\pi}} g(\boldsymbol{\pi})$, and

$$\boldsymbol{z}' = \operatorname{diag}^{-1}\left(\boldsymbol{B}\boldsymbol{A}^{-1}\boldsymbol{B}^T\right) = \sigma^{-2}(\operatorname{Var}_Q[s_j \,|\, \boldsymbol{y}]). \quad (5)$$

$\boldsymbol{z}'$ cannot be computed by a few LCG runs. Since $\boldsymbol{A}$ has no sparse graphical structure, we cannot use belief propagation either. However, the Lanczos algorithm can be used to estimate $\boldsymbol{z}'$ [10]. This algorithm is also essential for scoring many candidates in each design step of our method (see Section 3.1).

Our algorithm iterates between updates of $\boldsymbol{z}$ (outer loop steps) and inner loop convex optimization of $(\boldsymbol{u}, \boldsymbol{\pi})$. We show in [13] that $\min_{\boldsymbol{\pi}} \phi(\boldsymbol{\pi})$ is a *convex problem*, whenever all model sites are log-concave (as is the case for Laplacians), a finding which is novel to the best of our knowledge.

Once converged to the global optimum of $\phi(\boldsymbol{\pi})$, the posterior is approximated by $Q(\cdot|\boldsymbol{y})$ of (3), whose mean is given by $\boldsymbol{u}$. The main idea is to decouple $\phi(\boldsymbol{\pi})$ by upper bounding the critical term $\log|\boldsymbol{A}|$. If the $\boldsymbol{z}$ updates are done exactly, the algorithm is globally convergent [16]. Our algorithm is inspired by [16], where a different problem is addressed. Their method produces very sparse solutions of $\boldsymbol{X}\boldsymbol{u} \approx \boldsymbol{y}$, while our focus is on close approximate inference, especially w.r.t. the posterior covariance matrix. It was found in [12] that aggressive sparsification, notwithstanding being computationally convenient, *hurts* experimental design (and even reconstruction) for *natural images*. Their update of $\boldsymbol{z}$ requires (5) as well, but can be done more cheaply, since most $\pi_j = +\infty$, and $\boldsymbol{A}$ can be replaced by a much smaller matrix. Finally, note that MAP estimation [8] is solving (4) once for $\boldsymbol{z} = \boldsymbol{0}$, so can be seen as special case of our method.

### 3.1  Lanczos Algorithm. Efficient Design

The Lanczos algorithm [4] is typically used to find extremal eigenvectors of large, positive definite matrices $\boldsymbol{A}$. Requiring an MVM with $\boldsymbol{A}$ in each iteration, it produces $\boldsymbol{Q}^T\boldsymbol{A}\boldsymbol{Q} = \boldsymbol{T} \in \mathbb{R}^{k,k}$ after $k$ iterations, where $\boldsymbol{Q}^T\boldsymbol{Q} = \boldsymbol{I}$, $\boldsymbol{T}$ tridiagonal. Lanczos estimates of expressions linear in $\boldsymbol{\Sigma} = \boldsymbol{A}^{-1}$ are obtained by plugging in the low-rank approximation $\boldsymbol{Q}\boldsymbol{T}^{-1}\boldsymbol{Q}^T \approx \boldsymbol{\Sigma}$ [10]. In our case, $\boldsymbol{z}^{(k)} := \operatorname{diag}^{-1}(\boldsymbol{B}\boldsymbol{Q}\boldsymbol{T}^{-1}\boldsymbol{Q}^T\boldsymbol{B}^T) \to \boldsymbol{z}'$, $L^{(k)} := \log|\boldsymbol{T}| \to g(\boldsymbol{\pi})$. We also use Lanczos to compute entropy difference scores, approximating (2) by using $Q(\boldsymbol{u}|\boldsymbol{y})$ instead of $P(\boldsymbol{u}|\boldsymbol{y})$, and $Q'(\boldsymbol{u}|\boldsymbol{y}) \propto Q(\boldsymbol{u}|\boldsymbol{y})P(\boldsymbol{y}_*|\boldsymbol{u})$ instead of $P(\boldsymbol{u}|\boldsymbol{y}, \boldsymbol{y}_*)$, with $\boldsymbol{\pi}' = \boldsymbol{\pi}$. The expectation over $P(\boldsymbol{y}_*|\boldsymbol{y})$ need not be done then, and

$$\Delta(\boldsymbol{X}_*) \approx -\log|\boldsymbol{A}| + \log\left|\boldsymbol{A} + \boldsymbol{X}_*^T\boldsymbol{X}_*\right| = \log\left|\boldsymbol{I} + \boldsymbol{X}_*\boldsymbol{\Sigma}\boldsymbol{X}_*^T\right|.$$

For $nc$ candidates of $d$ rows, computing scores would need $d \cdot nc$ LCG runs, which is not feasible. Using the Lanczos approximation of $\boldsymbol{\Sigma}$, we need $k$ MVMs with $\boldsymbol{X}_*$ for each candidate, then $nc$ Cholesky decompositions of $\min\{k, d\} \times \min\{k, d\}$ matrices. Both computations can readily be parallelized, as is done in our implementation. Note that we can compute $\partial\Delta(\boldsymbol{X}_*)/\partial\alpha$ for $\boldsymbol{X}_* = \boldsymbol{X}_*(\alpha)$, if $\partial\boldsymbol{X}_*/\partial\alpha$ is known, so that gradient-based score optimization can be used.

The basic recurrence of the Lanczos method is treacherously simple. The loss of orthogonality in $\boldsymbol{Q}$ has to be countered, thus typical Lanczos codes are intricate. $\boldsymbol{Q}$ has to be maintained in memory. The matrices $\boldsymbol{A}$ we encounter here, have an almost *linearly* decaying spectrum, so standard Lanczos codes, designed for geometrically decaying spectra, have to be modified. Our $\boldsymbol{A}$ have no close low rank approximations, and eigenvalues from both ends of the spectrum converge rapidly in Lanczos. Therefore, our estimate $\boldsymbol{z}^{(k)}$ is *not* very close to the true $\boldsymbol{z}'$ even for quite large $k$. However, $\boldsymbol{z}^{(k)} \preceq \boldsymbol{z}'$, since $z_{k-1,j} \le z_{k,j}$ for all $j$. Since the sparsity penalty on $s_j$ in (4) is stronger for smaller $z_j$, underestimations from the Lanczos algorithm entail *more* sparsity (although still $z_{k,j} > 0$). In practice, a smaller $k$ often leads to somewhat *better* results, besides running much faster. While the global convergence proof for our algorithm hinges on exact updates of $\boldsymbol{z}$, which cannot be done to the best of our knowledge, the empirical success of Section 4 may be due to this observation, noting that natural image statistics are typically more super-Gaussian than the Laplacian. In conclusion, approximate inference requires the computation of marginal variances, which for general models cannot be approximated closely with generic techniques. In the context of sparse linear models, it seems to be sufficient to estimate the dominating covariance eigendirections, for which the Lanczos algorithm with a moderate number of steps can be used. More generally, the Lanczos method is a powerful tool for approximate inference in Gaussian models, an insight which does not seem to be widely known in machine learning.

## 4  Experiments

We start with some MRI terminology. An MR scanner acquires Fourier coefficients $Y(\boldsymbol{k})$ at spatial frequencies[5] $\boldsymbol{k}$ (the 2d Fourier domain is called $k$-*space*), along smooth *trajectories* $\boldsymbol{k}(t)$ determined by magnetic field gradients $\boldsymbol{g}(t)$. The control flow is called *sequence*. Its cost is determined by how long it takes to obtain a complete image, depending on the number of trajectories and their shapes. Gradient amplitude and slew rate constraints enforce smooth trajectories. In *Cartesian sampling*, trajectories are parallel equispaced lines in $k$-space, so the FFT can be used for image reconstruction. *Spiral sampling* offers a better coverage of $k$-space for given gradient power, leading to faster

acquisition. It is often used for dynamic studies, such as cardiac imaging and fMRI. A trajectory $\boldsymbol{k}(t)$ leads to data $\boldsymbol{y} = \boldsymbol{X_k}\boldsymbol{u}$, where $\boldsymbol{X_k} = [e^{-i2\pi \boldsymbol{r}_j^T \boldsymbol{k}(t_\ell)}]_{\ell j}$. We use gridding interpolation[6] with a Kaiser-Bessel kernel [1, ch. 13.2] to approximate the multiplication with $\boldsymbol{X_k}$, which would be too expensive otherwise. As for other reconstruction methods, most of our running time is spent in the gridding (MVMs with $\boldsymbol{X}$, $\boldsymbol{X}^T$, and $\boldsymbol{X}_*$).

For our experiments, we acquired data on an equispaced grid.[7] In theory, the image $\boldsymbol{u}$ is real-valued; in reality, due to resonance frequency offsets, magnetic field inhomogeneities, and eddy currents [1, ch. 13.4], the reconstruction contains a phase $\boldsymbol{\varphi}(\boldsymbol{r})$. It is common practice to discard $\boldsymbol{\varphi}$ after reconstruction. Short of modelling a complex-valued $\boldsymbol{u}$, we correct for low-frequency phase contributions by a cheap pre-measurement.[8] Note that $|\boldsymbol{u}_{true}|$, against which reconstructions are judged below, is not altered by this correction. From the

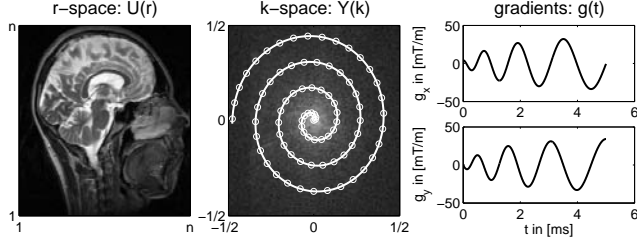

Figure 1: MR signal acquisition: $\boldsymbol{r}$-space and $k$-space representation of the signal on a rectangular grid as well as the trajectory obtained by means of magnetic field gradients

corrected raw data, we simulate all further measurements under different sequences using gridding interpolation. While no noise is added to these measurements, there remain significant high-frequency erroneous phase contributions in $\boldsymbol{u}_{true}$.

Interleaved outgoing *Archimedian spirals* employ trajectories $\boldsymbol{k}(t) \propto \theta(t)e^{i2\pi[\theta(t)+\theta_0]}$, $\theta(0) = 0$, where the gradient $\boldsymbol{g}(t) \propto d\boldsymbol{k}/dt$ grows to maximum strength at the slew rate, then stays there [1, ch. 17.6]. Sampling along an interleave respects the Nyquist limit. The number of revolutions $N_r$ and interleaves $N_{\text{shot}}$ determine the radial spacing. The scan time is proportional to $N_{\text{shot}}$. In our setup, $N_r = 8$, resulting in 3216 complex samples per interleave. For equispaced offset angles $\theta_0$, the *Nyquist spiral* (respecting the limit radially) has $N_{\text{shot}} = 16$. Our goal is to design spiral sequences with smaller $N_{\text{shot}}$, reducing scan time by a factor $16/N_{\text{shot}}$. We use the sequential method described in Section 2, where $\{\boldsymbol{X}_* \in \mathbb{R}^{n \times d}\}$ is a set of potential interleaves, $d = 6432$. The image resolution is $256 \times 256$, so $n = 65536$. Since $\boldsymbol{u}_{true}$ is approximately real-valued, measurements at $\boldsymbol{k}$ and $-\boldsymbol{k}$ are quite redundant, which is why we restrict[9] ourselves to offset angles $\theta_0 \in [0, \pi)$. We score candidates $(\pi/256)[0 : 255]$ in each round, comparing to equispaced placements $j\pi/N_{\text{shot}}$, and to drawing $\theta_0$ uniformly at random. For the former, favoured by MRI practitioners right now, the maximum $k$-space distance between samples is minimized, while the latter is aligned with compressed sensing recommendations [8].

For a given sequence, we consider different image reconstructions: the *posterior mode* (convex MAP estimation) [8], linear *least squares* (LS; linear conjugate gradients), and *zero filling with density compensation* (ZFDC; based on Voronoi diagram) [1, ch. 13.2.4]. The latter requires a single MVM with $\boldsymbol{X}^T$ only, and is most commonly used in practice. We selected the $\tau$ scale parameters (there are two of them, as in [12]) optimally for the Nyquist spiral $\boldsymbol{X}_{nyq}$, and set $\sigma^2$ to the variance of $\boldsymbol{X}_{nyq}(\boldsymbol{u}_{true} - |\boldsymbol{u}_{true}|)$. We worked on two slices (8,12) and used 750 Lanczos iterations in our method.[10] We report $L_2$ distances between reconstruction and true image $|\boldsymbol{u}_{true}|$. Results are given in Table 3, and some reconstructions (slice 8) are shown in Figure 2.

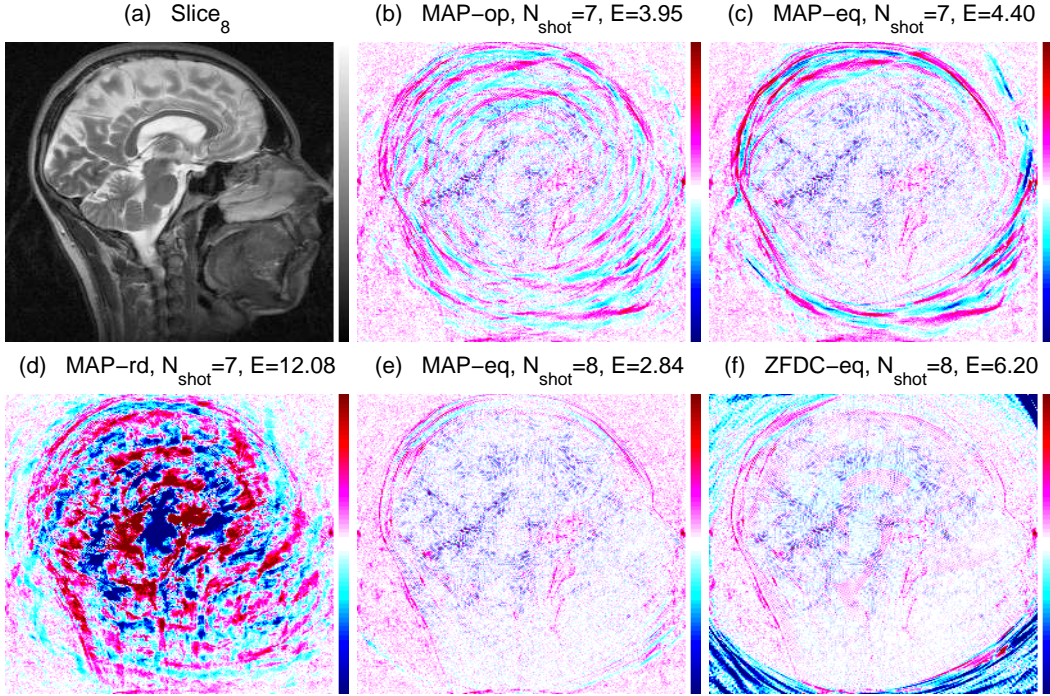

(a) Slice$_8$  (b) MAP−op, $N_{shot}$=7, E=3.95  (c) MAP−eq, $N_{shot}$=7, E=4.40

(d) MAP−rd, $N_{shot}$=7, E=12.08  (e) MAP−eq, $N_{shot}$=8, E=2.84  (f) ZFDC−eq, $N_{shot}$=8, E=6.20

Figure 2: Reconstruction results. Differences to true image (a; scale $[0, 1]$) in (b-f), scale $[-0.1, 0.1]$.

| $N_{shot}$ | img | MAP$_{op}$ | MAP$_{rd}$ | MAP$_{eq}$ | LS$_{op}$ | LS$_{rd}$ | LS$_{eq}$ | ZFDC$_{op}$ | ZFDC$_{rd}$ | ZFDC$_{eq}$ |
|---|---|---|---|---|---|---|---|---|---|---|
| 5 | 8 | 12.99 | 16.01 ± 2.49 | 14.18 | 17.23 | 19.97 ± 1.33 | 16.80 | 25.13 | 38.04 ± 6.14 | 23.51 |
| 6 | 8 | 8.31 | 12.46 ± 2.46 | 10.06 | 12.67 | 16.24 ± 1.13 | 13.19 | 18.79 | 33.29 ± 4.71 | 18.16 |
| 7 | 8 | 3.95 | 11.81 ± 2.71 | 4.40 | 7.80 | 13.71 ± 2.25 | 7.80 | 14.55 | 33.67 ± 5.90 | 12.73 |
| 8 | 8 | 2.94 | 6.86 ± 2.00 | 2.84 | 3.77 | 7.43 ± 2.48 | 3.31 | 13.08 | 26.96 ± 4.47 | 6.20 |
| 5 | 12 | 8.01 | 10.17 ± 1.63 | 9.32 | 12.77 | 14.95 ± 1.08 | 12.01 | 20.58 | 28.88 ± 4.25 | 19.74 |
| 6 | 12 | 4.94 | 7.74 ± 1.75 | 5.21 | 9.77 | 11.89 ± 0.95 | 9.77 | 16.33 | 25.47 ± 3.15 | 15.36 |
| 7 | 12 | 2.84 | 7.46 ± 1.80 | 3.18 | 6.40 | 9.95 ± 1.73 | 6.18 | 12.34 | 26.02 ± 3.44 | 10.62 |
| 8 | 12 | 2.20 | 4.60 ± 1.26 | 2.09 | 3.32 | 5.33 ± 1.73 | 2.27 | 10.07 | 21.47 ± 3.67 | 4.28 |

slices 2,4,6,10,12,14 from design of slice 8

| $N_{shot}$ | MAP$_{op}$ | MAP$_{eq}$ | LS$_{op}$ | LS$_{eq}$ |
|---|---|---|---|---|
| 5 | 9.01 ± 1.3 | 10.67 ± 2.1 | 14.70 ± 1.6 | 14.57 ± 2.1 |
| 6 | 5.43 ± 1.1 | 6.51 ± 2.1 | 10.80 ± 1.5 | 10.95 ± 1.8 |
| 7 | 3.00 ± 0.5 | 3.27 ± 0.8 | 7.08 ± 1.1 | 6.45 ± 1.4 |
| 8 | 2.42 ± 0.3 | 2.34 ± 0.3 | 3.16 ± 0.6 | 2.70 ± 0.6 |

| img | MAP$_{eq}$, $N_{shot} = 16$, (Nyq) | LS$_{eq}$, $N_{shot} = 16$, (Nyq) |
|---|---|---|
| 8 | 2.75 | 3.31 |
| 12 | 1.96 | 2.27 |

Figure 3: Results for spiral interleaves on slices 8, 12 (table left). Reconstruction: MAP (posterior mode [8]), LS (least squares), ZFDC (zero filling, density compensation). Offset angles $\theta_0 \in [0, \pi)$: op (optimized; our method), rd (uniformly random; avg. 10 runs), eq (equispaced). $N_{shot}$: Number of interleaves.
Table upper right: Avg. errors for slices 2,4,6,10,14, measured with sequences optimized on slice 8.
Table lower right: Results for Nyquist spiral eq[$N_{shot} = 16$].

The standard reconstruction method ZFDC is improved upon strongly by LS (both are linear, but LS is iterative), which in turn is improved upon significantly by MAP. This is true even for the Nyquist spiral ($N_{shot} = 16$). While the strongest errors of ZFDC lie outside the "effective field of view" (roughly circular for spiral), panel f of Figure 2 shows that ZFDC errors contain important structures all over the image. Modern implementations of LS and MAP are more expensive than ZFDC by moderate constant factors. Results such as ours, together with the availability of affordable high-performance digital computation, strongly motivate the transition away from direct signal processing reconstruction algorithms to modern iterative statistical estimators. Note that ZFDC (and, to a lesser extent, LS) copes best with equispaced designs, while MAP works best with optimized angles. This is because the optimized designs leave larger gaps in $k$-space (see Figure 4). Nonlinear estimators can interpolate across such gaps to some extent, using image sparsity priors. Methods like ZFDC merely interpolate locally in $k$-space, uninformed about image statistics, so that violations of the Nyquist limit anywhere necessarily translate into errors.

It is clearly evident that drawing the spiral offset angles at random does not work well, even if MAP reconstruction is used as in [8]. The ratio MAP$_{rd}$/MAP$_{op}$ in $L_2$ error is 1.23, 1.45, 2.99, 2.33 in Table 3, upper left. While both MAP$_{op}$ and MAP$_{eq}$ essentially attain Nyquist performance with $N_{shot} = 8$, MAP$_{rd}$ does not decrease to that level even with $N_{shot} = 16$ (not shown). Our

results strongly suggest that randomizing MR sequences is not a useful design principle.[11] Similar shortcomings of randomly drawn designs were reported in [12], in a more idealized setup. Reasons why CS theory as yet fails to guide measurement design for real images, are reviewed there, see also [15]. Beyond the rather bad *average* performance of random designs, the large variance across trials in Table 3 means that in practice, a randomized sequence scan is much like a gamble. The outcome of our Bayesian optimized design is stable, in that sequences found in several repetitions gave almost identical reconstruction performance.

The closest competitors in Table 3 are $\text{MAP}_{\text{op}}$ and $\text{MAP}_{\text{eq}}$. Since $\boldsymbol{u}_{true}$ is close to real, both attain close to Nyquist performance up from $N_{\text{shot}} = 8$. In the true undersampling regime $N_{\text{shot}} \in \{5, 6, 7\}$, $\text{MAP}_{\text{op}}$ improves significantly[12] upon $\text{MAP}_{\text{eq}}$. Comparing panels b,c of Figure 2, the artifact across the lower right leads to distortions in the mouth area. Undersampling artifacts are generally amplified by regular sampling, which is avoided in the optimized designs. Breaking up such regular designs seems to be the major role of randomization in CS theory, but our results show that much is lost in the process. We see that approximate Bayesian experimental design is useful to

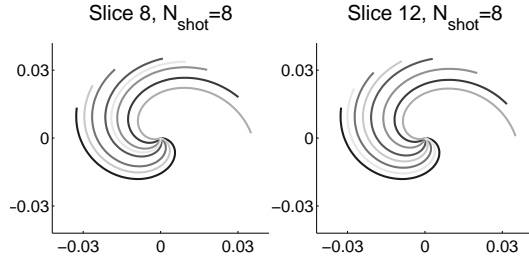

Figure 4: Spirals found by our algorithm. The ordering is color-coded: dark spirals selected first.

optimize measurement architectures for subsequent MAP reconstruction. To our knowledge, no similar design optimization method based purely on MAP estimation has been proposed (ours needs approximate inference), rendering the beneficial interplay between our framework and subsequent MAP estimation all the more interesting. The computational primitives required for MAP estimation and our method are the same. Our implementation requires about 5 hours on a single standard desktop machine to optimize 11 angles sequentially, 256 candidates per extension, with $n$ and $d$ as above. The score computations dominate the running time, but can readily be parallelized.

It is neither feasible nor desirable on most current MR scanners to optimize the sequence during the measurement, so an important question is whether sequences optimized on some slices work better in general as well (for the same contrast and similar objects). We tested transferability by measuring five other slices not seen by the optimization method. The results (Table 3, upper right) indicate that the main improvements are not specific to the object the sequence was optimized for.[13] Two spirals found by our method are shown in Figure 4 (2 of 8 interleaves, $N_{shot} = 8$). The spacing is not equidistant, and as noted above, only nonlinear MAP estimation can successfully interpolate across resulting larger $k$-space gaps. On the other hand, the spacing is more regular than is typically achieved by random sampling.

## 5    Discussion

We have presented the first scalable Bayesian experimental design framework for automatically optimizing MRI sequences, a problem of high impact on clinical diagnostics and brain research. The high demands on image resolution and processing time which come with this application are met in principle by our novel variational inference algorithm, reducing computations to signal processing

primitives such as FFT and gridding. We demonstrated the power of our approach in a study with spiral sequences, using raw data from a 3T MR scanner. The sequences found by our method lead to reconstructions of high quality, even though they are faster than traditionally used Nyquist setups by a factor up to two. They improve strongly on sequences obtained by blind randomization. Moreover, across all designs, nonlinear Bayesian MAP estimation was found to be essential for reconstructions from undersamplings, and our design optimization framework is especially useful for subsequent MAP reconstruction.

Our results strongly suggest that modifications to standard sequences can be found which produce similar images at lower cost. Namely, with so many handles to turn in sequence design nowadays, this is a high-dimensional optimization problem dealing with signals (images) of high complexity, and human experts can greatly benefit from goal-directed machine exploration. Randomizing parameters of a sequence, as suggested by compressed sensing theory, helps to break wasteful symmetries in regular standard sequences. As our results show, many of the advantages of regular sequences are lost by randomization though. The optimization of Bayesian information leads to irregular sequences as well, improving on regular, and especially on randomized designs. Our insights should be especially valuable in MR applications where a high temporal resolution is essential (such as fMRI studies), so that dense spatial sampling is not even an option. An extension to 3d volume reconstruction, making use of non-Gaussian hidden Markov models, is work in progress. Finally, our framework seems also promising for real-time imaging [1, ch. 11.4], where the scanner allows for on-line adaptations of the sequence depending on measurement feedback. It could be used to help an operator homing in on regions of interest, or could even run without human intervention.

We intend to test our proposal directly on an MR scanner, using the sequential setup described in Section 2. This will come with new problems not addressed in Section 4, such as phase or image errors that depend on the sequence employed[14] (which could be accounted for by a more elaborate noise model). In our experiments in Section 4, the choice of different offset angles is cost-neutral, but when a larger set of candidates is used, respective costs have to be quantified in terms of real scan time, error-proneness, heating due to rapid gradient switching, and other factors.

### Acknowledgments

We thank Stefan Kunis for help and support with NFFT.

## Footnotes

[1]These come from the presence of edges and smooth areas, which on a low level *define* image structure, and which are not present in Gaussian data (noise).

[2]The model we use has $q = 196096$ potentials and $n = 65536$ latent variables. Any algorithm that iterates over single potentials, has to solve at least $q$ linear systems of size $n$, while our method often converges after solving less than 50 of these.

[3]Phase contributions in $\boldsymbol{u}$ are discussed in Section 4.

[4]The end result is valid for singular $\boldsymbol{B}^T \boldsymbol{\Pi} \boldsymbol{B}$, by a continuity argument.

[5]Both $\boldsymbol{k}$ and spatial locations $\boldsymbol{r}$ are seen as $\in \mathbb{R}^2$ or $\in \mathbb{C}$.

[6]NFFT: http://www-user.tu-chemnitz.de/~potts/nfft/

[7]Field of view (FOV) 260mm ($256 \times 256$ voxels, $1\text{mm}^2$), 16 brain slices with a turbo-spin sequence, 23 echoes per excitation. Train of $120°$ refocusing pulses, each phase encoded differently. Slices are 4mm thick.

[8]We sample the center of $k$-space on a $p \times p$ Cartesian grid, obtaining a low-resolution reconstruction by FFT, whose phase $\tilde{\varphi}$ we use to correct the raw data. We tried $p \in \{16, 32, 64\}$ (larger $p$ means better correction), results below are for $p = 32$ only. While reconstruction errors generally decrease somewhat with larger $p$, the relative differences between all settings below are insensitive to $p$.

[9]Dropping this restriction disfavours equispaced $\{\theta_0\}$ setups with even $N_{\text{shot}}$.

[10]This seems small, given that $n = 65536$. We also tried 1250 iterations, which needed more memory, ran almost twice as long, and gave slightly *worse* results (see end of Section 3.1).

[11]Images exhibit a decay in power as function of spatial frequence (distance to $k$-space origin), and the most evident failure of uniform random sampling is the ignorance of this fact [15]. While this point is noted in [8], the variable-density weighting suggested there is built in to all designs compared here. *Any* spiral interleave samples more closely around the origin. In fact, the sampling density as a function of spatial frequency $|\boldsymbol{k}(t)|$ *does not depend* on the offset angles $\theta_0$.

[12]In another set of experiments (not shown), we compared optimization, randomization, and equispacing of $\theta_0 \in [0, 2\pi)$, in disregard of the approximate real-valuedness of $\boldsymbol{u}_{true}$. In this setting, equispacing performs poorly (worse than randomization).

[13]However, it is important that the object exhibits realistic natural image statistics. Artificial phantoms of extremely simple structure, often used in MR sequence design, are *not* suitable in that respect. Real MR images are much more complicated than simple phantoms, even in low level statistics, and results obtained on phantoms only should not be given overly high attendance.

[14]Some common problems with spirals are discussed in [1, ch. 17.6.3], together with remedies.

### References

[1] M.A. Bernstein, K.F. King, and X.J. Zhou. *Handbook of MRI Pulse Sequences*. Elsevier Academic Press, 1st edition, 2004.

[2] A. Garroway, P. Grannell, and P. Mansfield. Image formation in NMR by a selective irradiative pulse. *J. Phys. C: Solid State Phys.*, 7:L457–L462, 1974.

[3] M. Girolami. A variational method for learning sparse and overcomplete representations. *N. Comp.*, 13:2517–2532, 2001.

[4] G. Golub and C. Van Loan. *Matrix Computations*. Johns Hopkins University Press, 3rd edition, 1996.

[5] A. Haase, J. Frahm, D. Matthaei, W. Hänicke, and K. Merboldt. FLASH imaging: Rapid NMR imaging using low flip-angle pulses. *J. Magn. Reson.*, 67:258–266, 1986.

[6] J. Hennig, A. Nauerth, and H. Friedburg. RARE imaging: A fast imaging method for clinical MR. *Magn. Reson. Med.*, 3(6):823–833, 1986.

[7] P. Lauterbur. Image formation by induced local interactions: Examples employing nuclear magnetic resonance. *Nature*, 242:190–191, 1973.

[8] M. Lustig, D. Donoho, and J. Pauly. Sparse MRI: The application of compressed sensing for rapid MR imaging. *Magn. Reson. Med.*, 85(6):1182–1195, 2007.

[9] P. Mansfield. Multi-planar image formation using NMR spin-echoes. *J. Phys. C*, 10:L50–L58, 1977.

[10] M. Schneider and A. Willsky. Krylov subspace estimation. *SIAM J. Comp.*, 22(5):1840–1864, 2001.

[11] M. Seeger. Bayesian inference and optimal design for the sparse linear model. *JMLR*, 9:759–813, 2008.

[12] M. Seeger and H. Nickisch. Compressed sensing and Bayesian experimental design. In *ICML 25*, 2008.

[13] M. Seeger and H. Nickisch. Large scale variational inference and experimental design for sparse generalized linear models. Technical Report TR-175, Max Planck Institute for Biological Cybernetics, Tübingen, Germany, September 2008.

[14] M. Tipping and A. Faul. Fast marginal likelihood maximisation for sparse Bayesian models. In *AI and Statistics 9*, 2003.

[15] Y. Weiss, H. Chang, and W. Freeman. Learning compressed sensing. Snowbird Learning Workshop, Allerton, CA, 2007.

[16] D. Wipf and S. Nagarajan. A new view of automatic relevance determination. In *NIPS 20*, 2008.

